# Policy Gradient Coagent Networks

**Philip S. Thomas**
Department of Computer Science
University of Massachusetts Amherst
Amherst, MA 01002
pthomas@cs.umass.edu

## Abstract

We present a novel class of actor-critic algorithms for actors consisting of sets of interacting modules. We present, analyze theoretically, and empirically evaluate an update rule for each module, which requires only local information: the module's input, output, and the TD error broadcast by a critic. Such updates are necessary when computation of compatible features becomes prohibitively difficult and are also desirable to increase the biological plausibility of reinforcement learning methods.

## 1 Introduction

Methods for solving sequential decision problems with delayed reward, where the problems are formulated as Markov decision processes (MDPs), have been compared to the learning mechanisms of animal brains [3, 4, 9, 10, 13, 20, 22]. These comparisons stem from similarities between activation of dopaminergic neurons and reward prediction error [19], also called the temporal difference (TD) error [21]. Dopamine is broadcast to large portions of the human brain, suggesting that it may be used in a similar manner to the TD error in reinforcement learning (RL) [23] systems, i.e., to facilitate improvements to the brain's decision rules.

Systems with a *critic* that computes and broadcasts the TD error to another module called the *actor*, which stores the current decision rule, are called *actor-critic* architectures. Chang et al. [7] present a compelling argument that the fly brain is an actor-critic by finding the neurons making up the critic and then artificially activating them to train the actor portions of the brain. However, current actor-critic methods in the artificial intelligence community remain biologically implausible because each component of the actor can only be updated with detailed knowledge of the entire actor. This forces computational neuroscientists to either create novel methods [14] or alter existing methods from the artificial intelligence community in order to enforce locality constraints (e.g., [16]).

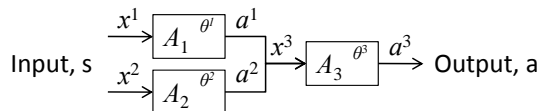

Figure 1: Example modular actor.

The actor in an actor-critic maintains a decision rule, $\pi$, called a *policy*, parameterized by a vector $\theta$, that computes the probability of an action (decision), $a$, given an estimate of the current state of the world, $s_t$, and the current parameters, $\theta_t$. In some cases, an actor can be broken into multiple interacting modules, each of which computes an action given some input, $x$, which may contain elements of $s$ as well as the outputs of other modules. An example of such a *modular actor* is provided in Figure 1. This actor consists of three modules, $A_1$, $A_2$, and $A_3$, with parameters $\theta^1, \theta^2,$

and $\theta^3$, respectively. The $i$th module takes input $x^i$, which is a subset of the state features and the outputs of other modules. It then produces its action $a^i$ according to its policy, $\pi^i(x^i, a^i, \theta^i) = \Pr(a^i|x^i, \theta^i)$. The output, $a$, of the whole modular actor is one of the module outputs—in this case $a = a^3$. Later we modify this to allow the action $a$ to follow any distribution with the state and module outputs as parameters. This modular policy can also be written as a non-modular policy that is a function of $\theta = \{\theta^1, \theta^2, \theta^3\}$, i.e., $\pi(s, a, \theta) = \Pr(a|s, \theta)$. We assume that the modular policy is not recurrent. Such modular policies appear frequently in models of the human brain, with modules corresponding to neurons or collections thereof [12, 16].

Current actor-critic methods (e.g. [11, 15, 23, 24]) require knowledge of $\partial\pi/\partial\theta^i$ in order to update $\theta^i$. However, $\partial\pi/\partial\theta^i$ often depends on the current values of all other parameters as well as the structure defining how the parameters are combined to produce the decision rule. This is akin to assuming that a neuron (or cluster of neurons), $A_i$, must know its influence on the final decision rule implemented. Were another module to modify its policy such that $\partial\pi/\partial\theta^i$ changes, a message must be sent to alert $A_i$ of the exact changes so that it can update its estimate of $\partial\pi/\partial\theta^i$, which is biologically implausible.

Rather than keeping a current estimate of $\partial\pi/\partial\theta^i$, one might attempt to compute it on the fly via the error backpropagation learning algorithm [17]. In this algorithm, each module, $A_i$, beginning with the output modules, computes its own update and then sends a message containing $\partial\pi/\partial a^j$ to each $A_j$ that $A_i$ uses as input (we call these $A_j$ *parents*, and $A_i$ a *child* of $A_j$). Once all of $A_i$'s children have updated, it will have all of the information required to compute $\partial\pi/\partial\theta^i$. Though an improvement upon the naive message passing scheme, backpropagation remains biologically implausible because it would require rapid transmission of information backwards along the axon, which has not been observed [8, 28]. However, gradient descent remains one of the most frequently used methods. For example, Rivest et al. [16] use gradient descent to update a modular actor, and are forced to assume that certain derivatives are always one in order to maintain realistic locality constraints.

This raises the question: could each module update given only local information that does not include explicit knowledge of $\partial\pi/\partial\theta^i$? We assume that a critic exists that broadcasts the TD error, so a module's local information would consist of its input $x^i$, which is not necessarily a Markov state representation, its output $a^i$, and the TD error. Though this has been achieved for tasks with immediate rewards [3, 26, 27], we are not aware of any such methods for tasks with delayed rewards. In this paper we present a class of algorithms, called *policy gradient coagent networks* (PGCNs), that do exactly this: they allow modules to update given only local information.

PGCNs are also a viable technique for non-biological reinforcement learning applications in which $\partial\pi/\partial\theta$ is prohibitively difficult to compute. For example, consider an artificial neural network where the output of each neuron follows some probability distribution over the reals. Though this would allow for exploration at every level, rather than just at the level of primitive actions of the output layer, expressions for $\pi(s, a, \theta)$ would require a nested integral for every node and $\partial\pi/\partial\theta$ would be difficult to compute or approximate for networks with many neurons and layers. Because PGCNs do not require knowledge of $\partial\pi/\partial\theta$, they remain simple even in such cases, making them a practical choice for complex parameterized policies.

## 2   Background

An MDP is a tuple $M = (\mathcal{S}, \mathcal{A}, \mathcal{P}, \mathcal{R}, d_{s_0})$, where $\mathcal{S}$ and $\mathcal{A}$ are the sets of possible states and actions respectively, $\mathcal{P}$ gives state transition probabilities: $\mathcal{P}(s, a, s') = \Pr(s_{t+1}{=}s'|s_t{=}s, a_t{=}a)$, where $t$ is the current time step, $\mathcal{R}(s, a) = E[r_t|s_t{=}s, a_t{=}a]$ is the expected reward when taking action $a$ in state $s$, and $d_{s_0}(s) = \Pr(s_0{=}s)$. An agent $A$ with time-variant parameters $\theta_t \in \Theta$ (typically function approximator weights, learning rates, etc.) observes the current state $s_t$, selects an action, $a_t$, based on $s_t$ and $\theta_t$, which is used to update the state according to $\mathcal{P}$. It then observes the resulting state, $s_{t+1}$, receives uniformly bounded reward $r_t$ according to $\mathcal{R}$, and updates its parameters to $\theta_{t+1}$.

A *policy* is a mapping from states to probabilities of selecting each possible action. $A$'s policy $\pi$ may be parameterized by a vector, $\theta$, such that $\pi(s, a, \theta) = \Pr(a_t{=}a|s_t{=}s, \theta_t{=}\theta)$. We assume that $\partial\pi(s, a, \theta)/\partial\theta$ exists for all $s, a,$ and $\theta$. Let $d_M^\theta(s)$ denote the stationary distribution over states

under the policy induced by $\theta$. We can then write the average reward for $\theta$ as

$$J_M(\theta) = \lim_{T \to \infty} \frac{1}{T} E \left[ \sum_{t=0}^{T-1} r_t \Big| M, \theta \right]. \tag{1}$$

The *state-value function*, which maps states to the difference between the average reward and the expected reward if the agent follows the policy induced by $\theta$ starting in the provided state, is

$$V_M^\theta(s) = \sum_{t=1}^{\infty} E[r_t - J(\theta)|M, s_0 = s, \theta]. \tag{2}$$

Lastly, we define the TD error to be $\delta_t = r_t - J_M(\theta) + V_M^\theta(s_{t+1}) - V_M^\theta(s_t)$.

## 2.1 Policy Gradient

One approach to improving a policy for an MDP is to adjust the parameters $\theta$ to ascend the *policy gradient*, $\nabla_\theta J_M(\theta)$. For reviews of policy gradient methods, see [5, 15, 24]. A common variable in policy gradient methods is the *compatible features*, $\psi_{sa} = \nabla_\theta \log \pi(s, a, \theta)$. Bhatnagar et al. [5] showed that $\delta_t \psi_{sa}$ is an unbiased estimate of $\nabla_\theta J_M(\theta)$ if $s \sim d_M^\theta(\cdot)$ and $a \sim \pi(s, \cdot, \theta)$. This results in a simple actor-critic algorithm, which we reproduce from [5]:

$$\hat{J}_{t+1} = (1 - c\alpha_t)\hat{J}_t + c\alpha_t r_t \tag{3}$$

$$\delta_t = r_t - \hat{J}_{t+1} + v_t \cdot \phi(s_{t+1}) - v_t \cdot \phi(s_t) \tag{4}$$

$$v_{t+1} = v_t + \alpha_t \delta_t \phi(s_t) \tag{5}$$

$$\theta_{t+1} = \theta_t + \beta_t \delta_t \psi_{s_t a_t}, \tag{6}$$

where $\hat{J}$ is a scalar estimate of $J$, $\delta_t$ remains the scalar TD error, $\phi$ is any function taking $\mathcal{S}$ to a feature space for linear value function approximation, $v$ is a vector of weights for the approximation $v \cdot \phi(s) \approx V_M^\theta(s)$, $c$ is a constant, and $\alpha_t$ and $\beta_t$ are learning rate schedules such that

$$\sum_{t=0}^{\infty} \alpha_t = \sum_{t=0}^{\infty} \beta_t = \infty, \sum_{t=0}^{\infty} (\alpha_t^2 + \beta_t^2) < \infty, \alpha_t = o(\beta_t). \tag{7}$$

One example of such a schedule would be $\alpha_t = \frac{\alpha \alpha_c}{\alpha_c + t^{2/3}}$ and $\beta_t = \frac{\beta \beta_C}{\beta_C + t}$, for some constants $\alpha, \alpha_C, \beta$, and $\beta_C$. We call this algorithm the *vanilla actor-critic* (VAC). Bhatnagar et al. [5] show that under certain mild assumptions and in the limit as $t \to \infty$, VAC will converge to a $\theta_t$ that is within a small neighborhood of a local maximum of $J_M(\theta)$.

Some more advanced actor-critic methods ascend the *natural policy gradient* [1, 5, 15],

$$\widetilde{\nabla}_\theta J_M(\theta) = \mathbf{G}(\theta)^{-1} \nabla_\theta J_M(\theta), \tag{8}$$

where $\mathbf{G}(\theta) = E_{s \sim d_M^\theta(\cdot), a \sim \pi(s, \cdot, \theta)}[\nabla_\theta \log \pi(s, a, \theta) \nabla_\theta \log \pi(s, a, \theta)^T]$ is the Fisher information matrix of the policy. To help differentiate between the two types of policy gradients, we refer to the non-natural policy gradient as the *vanilla policy gradient* hereafter. One view of the natural gradient is that it corrects for the skewing of the vanilla gradient that is induced by a particular parameterization of the policy [2]. Empirical studies have found that ascending the natural gradient results in faster convergence [1, 5, 15]. One algorithm for ascending the natural policy gradient is the **N**atural-**G**radient **A**ctor-**C**ritic with Advantage Parameters [5], which we abbreviate as NAC and use in our case study.

VAC and NAC have a property, which we reference later as Property 1, that is common to almost all other actor-critic methods: if the policy is a function of $x = f(s)$, for any $f$, such that $\pi(s, a, \theta)$ can be written as $\pi(x, a, \theta)$ or $\pi(f(s), a, \theta)$, then updates to the policy parameters $\theta$ are independent of $s$ given $x, a$, and $\delta_t$. For example, if $s = (s_1, s_2)$ and $f(s) = s_1$ so that the policy is a function of only $s_1$, then the update to $\theta$ requires knowledge of only $s_1, a$, and $\delta_t$, and not $s_2$. This is one crucial property will allow the actor to update given only local information.

VAC and NAC, as well as all other algorithms referenced, require computation of $\nabla_\theta \log \pi(s, a, \theta)$. Hence, none of these methods allow for local updates to modular policies, which makes them undesirable from a biological standpoint, and impractical for policies for which this derivative is prohibitively difficult to compute. However, by combining these methods with the CoMDP framework reviewed in Section 2.2 and by taking advantage of Property 1, the updates to the actor can be modified to satisfy the locality constraint.

## 2.2 Conjugate Markov Decision Processes

In this section we review the aspects of the conjugate Markov decision process (CoMDP) framework that are relevant to this work. Though Thomas and Barto [25] present the CoMDP framework for the discounted reward setting with finite state, action, and reward spaces, the extension to the average reward and infinite setting used here is straightforward. To solve $M$, one may create a network of agents $\mathbb{A}_1, \mathbb{A}_2, \ldots, \mathbb{A}_n$, where $\mathbb{A}_i$ has output $a^i \in \mathcal{A}^i$, where $\mathcal{A}^i$ is any space, though typically the reals or integers. All agents receive the same reward. We focus on the case where $\mathbb{A}_i = \{A_i, C_i\}$ are all actor critics, i.e., they contain an actor, $A_i$, and a critic, $C_i$. The action $a_t \in \mathcal{A}$ for $M$ is computed as $a_t \sim \Gamma(s, a^1, a^2, \ldots, a^n)$, for some distribution $\Gamma$. Each agent $\mathbb{A}_i$ has parameters $\theta^i$ defining its policy. We define $\bar{\theta}^i = \bigcup_{j \in \{1,2,\ldots,n\}-\{i\}} \theta^j$ to be the parameters of all agents other than $\mathbb{A}_i$. Each agent takes as input $s^i$, which contains the state of $M$ and the outputs of an arbitrary number of other agents: $s^i \in \mathcal{S} \times \prod_j \mathcal{A}^j$, where $\prod_j \mathcal{A}^j$ is the Cartesian product of the output sets of all the $\mathbb{A}_j$ whose output is used as input to $\mathbb{A}_i$. Notice that $s^i$ are not the components of $s$, but rather $s$ is the state of $M$, while $s^i$ is the input to $\mathbb{A}_i$. We require the graph with nodes for each $\mathbb{A}_i$ and a directed edge from $\mathbb{A}_i$ to $\mathbb{A}_j$ if $\mathbb{A}_j$ takes $a^i$ as part of its input, to be acyclic. Thus, the network of agents must be *feed-forward*, so we can assume an ordering of $\mathbb{A}_i$ such that if $a^j$ is part of $s^i$, then $j < i$. When executing the modular policy, the policies of the $\mathbb{A}_i$ can be executed in this order so that all requisite information for computing a module's output is always available. Thomas and Barto [25] call each $\mathbb{A}_i$ a *coagent* and the entire network a *coagent network*.

An agent $\mathbb{A}_i$ may treat the rest of the network and $M$ as its environment, where it sees states $s_t^i$ and takes actions $a_t^i$ resulting in reward $r_t$ (the same for all $\mathbb{A}_i$) and a transition to state $s_{t+1}^i$. This environment is called a *conjugate Markov decision process* (CoMDP), which is an MDP $M^i = (\mathcal{S} \times \prod_j \mathcal{A}^j, \mathcal{A}^i, \mathcal{P}^i, \mathcal{R}^i, d_{s_0}^i)$ where $\mathcal{S} \times \prod_j \mathcal{A}^j$ is the state space, $\mathcal{A}^i$ is the action space, $\mathcal{P}^i(s^i, a^i, \hat{s}^i) = \Pr\{s_{t+1}^i = \hat{s}^i | s_t^i = s^i, a_t^i = a^i, M, \bar{\theta}^i\}$, $\mathcal{R}^i(s^i, a^i) = E[r_t | s_t^i = s^i, a_t^i = a^i, M, \bar{\theta}^i]$ gives the expected reward when taking action $a$ in state $s$, and $d_{s_0}^i$ is the distribution over initial states of $M^i$. We write $\pi^i(s^i, a^i, \theta^i)$ to denote $\mathbb{A}_i$'s policy for $M^i$. Notice that $M^i$ depends on $\bar{\theta}^i$. Thus, as the policies of other coagents change, so too does the CoMDP with which $\mathbb{A}_i$ interacts. While [25] considers generic methods for handling this nonstationarity, we focus on the special case in which all $\mathbb{A}_i$ are policy gradient methods.

Theorem 3 of [25] states that the policy gradient of $M$ can be decomposed into the policy gradients for all of the CoMDPs, $M^i$:

$$\frac{\partial J_M(\theta^1, \theta^2, \ldots, \theta^n)}{\partial[\theta^1, \theta^2, \ldots, \theta^n]} = \left[ \frac{\partial J_M(\theta^1, \theta^2, \ldots, \theta^n)}{\partial \theta^1}, \frac{\partial J_M(\theta^1, \theta^2, \ldots, \theta^n)}{\partial \theta^2}, \ldots, \frac{\partial J_M(\theta^1, \theta^2, \ldots, \theta^n)}{\partial \theta^n} \right]$$

$$= \left[ \frac{\partial J_{M^1}(\theta^1)}{\partial \theta^1}, \frac{\partial J_{M^2}(\theta^2)}{\partial \theta^2}, \ldots, \frac{\partial J_{M^n}(\theta^n)}{\partial \theta^n} \right]. \tag{9}$$

Thus, if each coagent computes and follows the policy gradient based on the local environment that it sees, the coagent network will follow its policy gradient on $M$.

Thomas and Barto [25] also show that the value functions for $M$ and all the CoMDPs are the same for all $s_t$, if the additional state components of $M^i$ are drawn according to the modular policy:

$$V_{M^1}^{\theta^1}(s_t^1) = V_{M^2}^{\theta^2}(s_t^2) = \ldots = V_{M^n}^{\theta^n}(s_t^n) = V_M^{\theta}(s_t). \tag{10}$$

The state-value based TD error is therefore the same as well:

$$\delta_t = r_t - J_M(\theta) + V_M^{\theta}(s_{t+1}) - V_M^{\theta}(s_t) = r_t - J_{M^i}(\theta^i) + V_{M^i}^{\theta^i}(s_{t+1}^i) - V_{M^i}^{\theta^i}(s_t^i), \forall i. \tag{11}$$

This means that, if the coagents require $\delta_t$, we can maintain a *global critic*, $C$, that keeps an estimate of $V_M^{\theta}$, which can be used to replace every $C_i$ by computing $\delta_t$ and broadcasting it to each $A_i$. Because all $\mathbb{A}_i$ share a global critic, $C$, all that remains of each module is the actor $A_i$. We therefore refer to each $A_i$ as a module.

Notice that the CoMDPs, $M^i$, and thus the coagents, $\mathbb{A}_i$, have $\mathcal{S}$ as part of their state space. This is required for $M^i$ to remain Markov. However, if the actor's policy is a function of some $x^i = f(s^i)$ for any $f$, i.e., the policy can be written as $\pi^i(x^i, a^i, \theta^i)$, then, by Property 1, updates to the actor's policy require only the TD error, $a_i$, and $x^i$. Hence, the full Markovian state representation is only needed by the global critic, $C$. The modules, $A_i$, will be able to perform their updates given only their input: the $x^i$ portion of the state of $M^i$.

## 3   Methods

The CoMDP framework tells us that, if each module is an actor that computes the policy gradient for its local environment (CoMDP), then the entire modular actor will ascend its policy gradient. Actor-critics satisfying Property 1 are able to perform their policy updates given only local information: the policy's input $x_t$, the most recent action $a_t$, and the TD error $\delta_t$. Combining these two, each module $A_i$ can compute its update given only its local input $x_t^i$, most recent action $a_t^i$, and the TD error $\delta_t$. We call any network of coagents, each using policy gradient methods, a *policy gradient coagent network* (PGCN). One PGCN is the *vanilla coagent network* (VCN), which uses VAC for all modules (coagents), and maintains a global critic that computes and broadcasts $\delta_t$. The VCN algorithm is depicted diagramatically in Figure 2, where $\psi_{x^i,a^i}^i = \nabla_{\theta^i} \log \pi^i(x^i, a^i, \theta^i)$ are the compatible features for the $i$th module. Notice that $\delta_t \psi_{x_t^i a_t^i}^i$ is an unbiased estimate of the policy gradient for $M^i$ [5], which is an unbiased estimate of part of the policy gradient for $M$ by Equation 9.

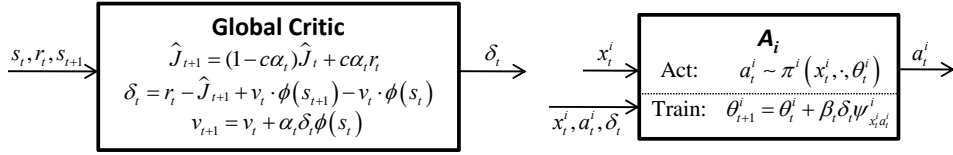

Figure 2: Diagram of the vanilla coagent network (VCN) algorithm. The global critic observes $s_t, r_t, s_{t+1}$ tuples, updates its estimate $\hat{J}$ of the average reward, which it uses to compute the TD error $\delta_t$, which is then broadcast to all of the modules, $A_i$. Lastly, it updates the parameters, $v$, of its state-value estimate. Each module $A_i$ draws its actions from $\pi^i(x_t^i, \cdot, \theta_t^i)$ and then computes updates to $\theta^i$ given its input $x_t^i$, action $a_t^i$, and the TD error, $\delta_t$, which was broadcast by the global critic.

To implement VCN, observe the current state $s_t$, compute the module outputs $a_t^i$ and then $a_t = \Gamma(s_t, a_t^1, a_t^2, \dots, a_t^n)$. This action will result in a transition to $s_{t+1}$ with reward $r_t$. Given $s_t, r_t$, and $s_{t+1}$ the global critic can execute to produce $\delta_t$, which can then be used to train each module $A_i$. Notice that the $A_i$ can update concurrently. This process then repeats.

## 4   The Decomposed Natural Policy Gradient

Another interesting PGCN, which we call a *natural coagent network* (NCN), would use coagents that ascend the natural policy gradient, e.g., NAC. However, Equation 9 does not hold for natural gradients:

$$\widetilde{\nabla}_\theta J_M(\theta) \neq \left[ \widetilde{\nabla}_{\theta^1} J_{M^1}(\theta^1), \widetilde{\nabla}_{\theta^2} J_{M^2}(\theta^2), \dots, \widetilde{\nabla}_{\theta^n} J_{M^n}(\theta^n) \right] \equiv \widehat{\nabla}_\theta J_M(\theta), \tag{12}$$

where $\theta = \left\{ \theta^1, \theta^2, \dots, \theta^n \right\}$ and $\widehat{\nabla}_\theta J_M(\theta)$ is an estimate of the natural policy gradient that we call the *decomposed natural policy gradient*, which has an implicit dependence on how $\theta$ is partitioned into $n$ components. Hence, a PGCN, where each module computes its natural policy gradient, would not follow the natural policy gradient, but rather $\widehat{\nabla}_\theta J_M(\theta) = \widehat{\mathbf{G}}(\theta)^{-1} \nabla_\theta J_M(\theta)$, an approximation thereto, where $\widehat{\mathbf{G}}(\theta)$ is an approximation of $\mathbf{G}(\theta)$, constructed by:

$$\widehat{\mathbf{G}}(\theta)_{ij} = \begin{cases} 0 & \text{if the } i \text{ and } j \text{th elements of } \theta \text{ are in different modules} \\ \mathbf{G}(\theta^k)_{ij} & \text{if the } i \text{ and } j \text{th elements of } \theta \text{ are both in module } A_k \end{cases}, \tag{13}$$

where $\mathbf{G}(\theta^k)$ is the Fisher information matrix of the $k$th module's policy:

$$\mathbf{G}(\theta^k) = E_{s^k \sim d_{M^k}^{\theta^k}(\cdot), a^k \sim \pi^k(x^k, \cdot, \theta^k)} [\nabla_{\theta^k} \log \pi^k(x^k, a^k, \theta^k) \nabla_{\theta^k} \log \pi^k(x^k, a^k, \theta^k)^T], \tag{14}$$

where $\mathbf{G}(\theta^k)_{ij}$ in Equation 13 denotes the entry corresponding to the $i$ and $j$th elements of $\theta$, which are elements of $\theta^k$.

The decomposed natural policy gradient is intuitively a trade-off between the natural policy gradient and the vanilla policy gradient depending on the granularity of modularization. For example, if the

policy is one module, $A_1$, and $\Gamma(s, a^1) = a^1$, then the decomposed natural policy gradient is trivially the same as the natural policy gradient. On the other hand, as the policy is broken into more and more modules, the gradient begins to differ more and more from the natural policy gradient, because the structure of the modular policy begins to influence the direction of the gradient. With finer granularity, $\widehat{\mathbf{G}}(\theta)$ will tend to a diagonal approximation of the identity matrix. If the modular actor contains one parameter per module and the module inputs are normalized, it is possible for $\widehat{\mathbf{G}}(\theta)^{-1} = \mathbf{I}$, in which case the decomposed natural policy gradient will be equivalent to the vanilla policy gradient. Hence, the more coarse the modularization (fewer modules), the closer the decomposed natural policy gradient is to the natural policy gradient, while the finer the modularization (more modules), the closer the decomposed natural policy gradient may come to the vanilla policy gradient.

Each term of the decomposed natural policy gradient is within ninety degrees of the vanilla policy gradient, so a system will converge to a local optimum if it follows the decomposed natural policy gradient and the step size is decayed appropriately.

## 5    Variance of Gradient Estimates

Let $\psi_{s,a,i} = \nabla_{\theta^i} \log \pi(s, a, \theta)$ be the components of $\psi_{s,a}$ that correspond to the parameters of $A_i$. Both $\delta_t \psi^i_{x^i, a^i}$, the update to the parameters of $A_i$ by VCN, and $\delta_t \psi_{s,a,i}$, the update by VAC, are unbiased estimates of $\nabla_{\theta^i} J_{M^i}(\theta^i) = \nabla_{\theta^i} J_M(\theta)$. This means that $E[\delta_t \psi_{s,a,i}] = E[\delta_t \psi^i_{x^i, a^i}]$, which is particularly interesting because $\delta_t$ is the same for both, so the only difference between the two are the compatible features used. Whereas $\psi_{s,a,i}$ requires computation of the derivative of the entire modular policy, $\pi$, $\psi^i_{x^i, a^i}$ only requires differentiation of $\pi^i$. Thus, the latter satisfies the locality constraint, and is also easier to compute. However, this benefit comes at the cost of higher variance.

This increase in variance appears regardless of the actor-critic method used. In this section we focus on VAC due to its simplicity, though the argument that stochasticity in the CoMDP is the root cause of the variance of gradient estimates carries over to PGCNs using other actor-critic methods as well. This increase in variance has also been observed in multi-agent reinforcement learning research as additional stochasticity in one agent's environment when another explores [18].

Consider using VAC on any MDP. Bhatnagar et al. [5] show that $E[\delta_t | s_t = s, a_t = a, M, \theta]$ can be viewed as the advantage of taking action $a_t$ in state $s_t$ over following the policy induced by $\theta$. If it is positive, it means taking $a_t$ in $s_t$ is better than following $\pi$. If it is negative, then $a_t$ is worse. So, following $E[\delta_t \psi_{s_t, a_t}]$ increases the likelihood of $a_t$ if it is advantageous, and decreases the likelihood of $a_t$ if it is disadvantageous. However, our updates use samples rather than the expected value, so an action $a_t$ that is actually worse could, due to stochasticity in the environment, result in a TD error that suggests it is advantageous. Thus, the gradient estimates are influenced by the stochasticity of the transition function $\mathcal{P}$ and reward function $\mathcal{R}$. If $\mathcal{P}$ or $\mathcal{R}$ is very stochastic, the same $s, a$ pair will result in seemingly random TD errors, which manifests as large variance in $\delta_t \psi_{s_t, a_t}$ samples.

Now consider the stochasticity in $M$ and $M^i$. The state transitions of $M^i$ depend not only on $M$'s transition function, but may also depend on the actions selected by some or all $A_j$, $j \neq i$. Consider the modular actor from Figure 1 in the case where the transitions and rewards of $M$ are deterministic. The transition function for $M^3$, the CoMDP for $A_3$, remains relatively deterministic because its actions completely determine the transitions of $M$. We therefore expect the variance in the gradient estimate for the parameters of $A_3$ to be only slightly higher for VCN than it is for VAC. However, the actions of $A_1$ and $A_2$ influence the transitions of $M$ indirectly through the actions of $A_3$, which adds a layer of stochasticity to their transition functions. We therefore expect policy gradient estimates for their parameters to have higher variance. In summary, the stochasticity in the CoMDPs is responsible for VCN's policy gradient estimates having higher variance than those of VAC.

We performed a simple study using the modular actor from Figure 1 on a $10 \times 10$ gridworld with deterministic actions $\{up, down, left, right\}$, a reward of $-1$ for all transitions, factored state $(\bar{x}, \bar{y})$, and with a terminal state at $(10, 10)$. For the modular actor, $\mathcal{A}^1 = \mathcal{A}^2 = \{0, 1\}$, $\mathcal{A}^3 = \{up, down, left, right\}$, $A_1$ and $A_2$ both received the full state $(\bar{x}, \bar{y})$, and all modules used

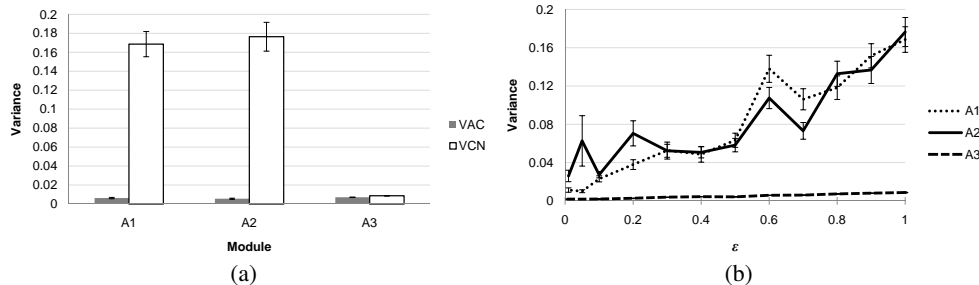

Figure 3: (a) Variance of the VAC and VCN updates for weights in each of the three modules. (b) Variance of updates using VCN with various $\varepsilon$. Standard error bars are provided ($n = 100$).

linear function approximation rather than a tabular state representation. All modules also used soft-max action selection:

$$\pi^i(x^i, a, \theta^i) = \frac{e^{\tau \theta^i_a \cdot x^i}}{\sum_{\hat{a} \in \mathcal{A}^i} e^{\tau \theta^i_{\hat{a}} \cdot x^i}}, \tag{15}$$

where $\tau$ is a constant scaling the amount of exploration, and where the parameters $\theta^i$ for the $i$th module contain a weight vector $\theta^i_a$ for each action $a \in \mathcal{A}^i$. The critic is common to both methods, and our goal is not to compare methods for value function approximation, so we used a tabular critic. With all actor weights fixed and selected randomly with uniform distribution from $(-1, 1)$, we first observed that the mean of the updates $\delta_t \psi_{s_t, a_t, i}$ and $\delta_t \psi^i_{x^i, a^i}$ are approximately equal, as expected, and then computed the variance of both updates. The results are shown in Figure 3(a). As predicted, the variance of the gradient estimates for each parameter of $A_1$ and $A_2$ is larger for VCN, though the variance of the gradient estimate for each parameter of $A_3$ is similar for VCN and VAC.

## 6 Variance Mitigation

To mitigate the increase in the variance of gradient estimates, we observe that, in general, the additional variance due to the other modules can be completely removed for a module $A_i$ if every other module is made to be deterministic. This is not practical because every module must explore in order to learn. However, we can approximate it by decreasing the exploration of each module, making its policy less stochastic and more greedy. For example, every module could take a deterministic greedy action without performing any updates with probability $1 - \varepsilon$ for some $\varepsilon \in [0, 1)$. With probability $\varepsilon$ the module would act using softmax action selection and update its parameters. As $\varepsilon \to 0$, the probability of two modules exploring simultaneously goes to zero, decreasing the variance in $M^i$ but also decreasing the percent of time steps during which each module trains. When $\varepsilon = 1$, every module explores and updates on every step, so the algorithm is the original PGCN algorithm (VCN if using VAC for each module).

We repeated the gridworld study of the variance in gradient estimates for various $\varepsilon$. The results, shown in Figure 3(b), show that smaller $\varepsilon$ can be effective in reducing the variance of gradient estimates. Notice that VCN using $\varepsilon = 1$ is equivalent to VCN as described previously, so the points for $\varepsilon = 1$ in Figure 3(b) correspond exactly to the VCN data in Figure 3(a). Thus, if the variance in gradient estimates precludes learning, we suggest making the policies of the modules more deterministic by decreasing exploration and increasing exploitation.

Several questions remain. First, though the variance decreases, the amount of exploration also decreases, so what is the net effect on learning speed? Second, how does PGCN compare to an actor-critic where $\nabla_\theta \pi(s, a, \theta)$ is known? Lastly, is there a significant loss in performance when using the decomposed natural policy gradient as opposed to the true natural policy gradient? We attempt to answer these questions in the following section.

| Algorithm | $\alpha$ | $\beta$ | $c$ | $\tau_{12}$ | $\tau_3$ | Average Reward | Standard Error |
|---|---|---|---|---|---|---|---|
| VAC | 0.75 | 0.25 | 0.13 | 0.5 | 2.5 | $-23.13$ | 0.09 |
| VCN | 0.25 | 0.1 | 0.04 | 0.1 | 3.5 | $-29.15$ | 0.09 |
| NAC | 0.5 | 0.1 | 0.02 | 0.05 | 1 | $-24.91$ | 0.08 |
| NCN | 0.5 | 0.1 | 0.02 | 0.05 | 1 | $-28.32$ | 0.14 |

Table 1: Best parameters found for each algorithm. The average reward per episode and standard error are computed using 10000 samples (each a lifetime of 75 episodes). The optimization tested each parameter set for 300 lifetimes, so the best parameters found still occasionally perform poorly. We found the above parameters to perform poorly (average reward less than $-200$) approximately one in 500 lifetimes. These outliers were removed for the average reward calculations. Random policy parameters average less than $-5000$ reward per episode.

## 7 Case Study

In this section we compare the learning speed of VAC, VCN, NAC, and NCN. Our goal is to determine whether VCN and NCN perform similarly to VAC and NAC, which are established methods [6], even though VCN and NCN's modules do not have access to $\partial \pi / \partial \theta^i$. To perform a thorough analysis, we again use the modular actor depicted in Figure 1, as in Section 5. We therefore require a problem with a simple optimal policy. We select the gridworld from Section 5, and again use a tabular critic in order to focus on the difference in policy improvements. To decrease the size of the parameter space, we did not decay $\alpha$ nor $\beta$. For all four algorithms, we performed a grid search for the $\alpha, \beta, c, \tau_{12}$, and $\tau_3$ that maximize the average reward over 75 episodes, where $\tau_{12}$ is the $\tau$ used by $A_1$ and $A_2$, while $\tau_3$ is that of $A_3$. The best parameters are provided in Table 1. Recall that the increased variance in VCN updates arises because $A_1$ and $A_2$'s actions only influence the transitions of $M$ indirectly through the actions of $A_3$. Though decreased exploration is beneficial in general, for this particular modular policy it is therefore particularly important that $A_3$'s exploration be decreased by increasing $\tau_3$. The optimization does just this, balancing the trade-off between exploration and the variance of gradient estimates by selecting larger $\tau_3$ for VCN than VAC. The mean ratio $\tau_3 / \tau_{12}$ for the top 25 of the 202300 parameters tested was 5.48 for VAC and 31.04 for VCN, further emphasizing the relatively smaller exploration of $A_3$. For NAC and NCN, the exploration parameters are identical, suggesting that the additional variance of gradient estimates was not significant. This is likely due to the policy gradient estimates being filtered before being used.

The average rewards during a lifetime are similar, suggesting that, even though the variance of gradient estimates can be orders larger for VCN with $\tau_{12} = \tau_3 = 1$ (Figure 3(a)), exploration can be tuned such that learning speed is not significantly diminished.

## 8 Conclusion

We have devised a class of algorithms, policy gradient coagent networks (PGCNs), and two specific instantiations thereof, the natural coagent network (NCN) and vanilla coagent network (VCN), which allow modules within an actor to update given only local information. We show that the NCN ascends the decomposed natural policy gradient, an approximation to the natural policy gradient, while VCN ascends the vanilla policy gradient. We discussed the theoretical properties of both the decomposed natural policy gradient and the increase in the variance of gradient estimates when using PGCNs. Lastly, we presented a case study to compare NCN and VCN to two existing actor-critic methods, NAC and VAC. We showed that, even though NAC and VAC are provided with additional non-local information, VCN and NCN perform comparably. We point out how VCN's similar performance is achieved by decreasing exploration in order to decrease the stochasticity of each module's CoMDP, and thus the variance of the gradient estimates.

## Acknowledgements

We would like to thank Scott Kuindersma, Scott Niekum, Bruno Castro da Silva, Andrew Barto, Sridhar Mahadevan, the members of the Autonomous Learning Laboratory, and the reviewers for their feedback and contributions to this paper.

# References

[1] S. Amari. Natural gradient works efficiently in learning. *Neural Computation*, 10(2):251–276, 1998.

[2] S. Amari and S. Douglas. Why natural gradient? In *Proceedings of the 1998 IEEE International Conference on Acoustics, Speech, and Signal Processing (ICASSP '98)*, volume 2, pages 1213–1216, 1998.

[3] A. G. Barto. Learning by statistical cooperation of self-interested neuron-like computing elements. *Human Neurobiology*, 4:229–256, 1985.

[4] A. G. Barto. Adaptive critics and the basal ganglia. *Models of Information Processing in the Basal Ganglia*, pages 215–232, 1995.

[5] S. Bhatnagar, R. S. Sutton, M. Ghavamzadeh, and M. Lee. Natural actor-critic algorithms. *Automatica*, 45(11):2471–2482, 2009.

[6] S. Bhatnagar, R. S. Sutton, M. Ghavamzadeh, and M. Lee. Natural actor-critic algorithms. Technical Report TR09-10, University of Alberta Department of Computing Science, June 2009.

[7] A. Claridge-Chang, R. Roorda, E. Vrontou, L. Sjulson, H. Li, J. Hirsh, and G. Miesenbock. Writing memories with light-addressable reinforcement circuitry. *Cell*, 193(2):405–415, 2009.

[8] F. H. C. Crick. The recent excitement about neural networks. *Nature*, 337:129–132, 1989.

[9] N. Daw and K. Doya. The computational neurobiology of learning and reward. *Current Opinion in Neurobiology*, 16:199–204, 2006.

[10] K. Doya. What are the computations of the cerebellum, the basal ganglia and the cerebral cortex? *Neural Networks*, 12:961–974, 1999.

[11] K. Doya. Reinforcement learning in continuous time and space. *Neural Computation*, 12(1):219–245, 2000.

[12] M. J. Frank and E. D. Claus. Anatomy of a decision: Striato-orbitofrontal interactions in reinforcement learning, decision making, and reversal. *Psychological Review*, 113(2):300–326, 2006.

[13] E. Ludvig, R. Sutton, and E. Kehoe. Stimulus representation and the timing of reward-prediction errors in models of the dopamine system. *Neural Computation*, 20:3034–3035, 2008.

[14] R. C. O'Reilly. *The LEABRA model of neural interactions and learning in the neocortex*. PhD thesis, Carnegie Mellon University.

[15] J. Peters and S. Schaal. Natural actor critic. *Neurocomputing*, 71:1180–1190, 2008.

[16] F. Rivest, Y. Bengio, and J. Kalaska. Brain inspired reinforcement learning. In *Advances in Neural Information Processing Systems*, pages 1129–1136, 2005.

[17] D. E. Rumelhart and J. L. McClelland. *Parallel distributed processing. Volume 1: Foundations*. MIT Press, Cambridge, MA, 1986.

[18] T. W. Sandholm and R. H. Crites. Multiagent reinforcement learning in the iterated prisoner's dilemma. *Biosystems*, 37:147–166, 1996.

[19] W. Schultz, P. Dayan, and P. Montague. A neural substrate of prediction and reward. *Science*, 275:1593–1599, 1992.

[20] A. Stocco, C. Lebiere, and J. Anderson. Conditional routing of information to the cortex: A model of the basal ganglia's role in cognitive coordination. *Psychological Review*, 117(2):541–574, 2010.

[21] R. Sutton. Learning to predict by the methods of temporal differences. *Machine Learning*, 3:9–44, 1988.

[22] R. Sutton and A. Barto. Toward a modern theory of adaptive networks: Expectation and prediction. *Psychological Review*, 88:135–140, 1981.

[23] R. Sutton and A. Barto. *Reinforcement learning: An introduction*. MIT Press, Cambridge, MA, 1998.

[24] R. Sutton, D. McAllester, S. Singh, and Y. Mansour. Policy gradient methods for reinforcement learning with function approximation. In *Advances in Neural Information Processing Systems 12*, pages 1057–1063, 2000.

[25] P. Thomas and A. Barto. Conjugate Markov decision processes. In *Proceedings of the Twenty-Eighth International Conference on Machine Learning*, 2011.

[26] R. J. Williams. A class of gradient-estimating algorithms for reinforcement learning in neural networks. In *Proceedings of the IEEE First International Conference on Neural Networks*, 1987.

[27] R. J. Williams. Simple statistical gradient-following algorithms for connectionist reinforcement learning. *Machine Learning*, 8(3):229–256, 1992.

[28] D. Zipser and R. A. Andersen. A back propagation programmed network that simulates response properties of a subset of posterior parietal neurons. *Nature*, 331:679–684, 1988.

